# An interior-point stochastic approximation method and an L1-regularized delta rule

**Peter Carbonetto**
pcarbo@cs.ubc.ca

**Mark Schmidt**
schmidtm@cs.ubc.ca

**Nando de Freitas**
nando@cs.ubc.ca

**Department of Computer Science**
University of British Columbia
Vancouver, B.C., Canada V6T 1Z4

## Abstract

The stochastic approximation method is behind the solution to many important, actively-studied problems in machine learning. Despite its far-reaching application, there is almost no work on applying stochastic approximation to learning problems with general constraints. The reason for this, we hypothesize, is that no robust, widely-applicable stochastic approximation method exists for handling such problems. We propose that *interior-point methods* are a natural solution. We establish the stability of a stochastic interior-point approximation method both analytically and empirically, and demonstrate its utility by deriving an on-line learning algorithm that also performs feature selection via $L_1$ regularization.

## 1  Introduction

The stochastic approximation method supplies the theoretical underpinnings behind many well-studied algorithms in machine learning, notably policy gradient and temporal differences for reinforcement learning, inference for tracking and filtering, on-line learning [1, 17, 19], regret minimization in repeated games, and parameter estimation in probabilistic graphical models, including expectation maximization (EM) and the contrastive divergences algorithm. The main idea behind stochastic approximation is simple yet profound. It is simple because it is only a slight modification to the most basic optimization method, gradient descent. It is profound because it suggests a fundamentally different way of optimizing a problem—instead of insisting on making progress toward the solution at every iteration, it only requires that progress be achieved *on average*.

Despite its successes, people tend to steer clear of constraints on the parameters. While there is a sizable body of work on treating constraints by extending established optimization techniques to the stochastic setting, such as projection [14], subgradient (e.g. [19, 27]) and penalty methods [11, 24], existing methods are either unreliable or suited only to specific types of constraints. We argue that a reliable stochastic approximation method that handles constraints is needed because constraints routinely arise in the mathematical formulation of learning problems, and the alternative approach—penalization—is often unsatisfactory.

Our main contribution is a new stochastic approximation method in which each step is the solution to the primal-dual system arising in interior-point methods [7]. Our method is easy to implement, dominates other approaches, and provides a general solution to constrained learning problems. Moreover, we show interior-point methods are remarkably well-suited to stochastic approximation, a result that is far from trivial when one considers that stochastic algorithms do not behave like their deterministic counterparts (e.g. Wolfe conditions [13] do not apply). We derive a variant of Widrow and Hoff's classic "delta rule" for on-line learning (Sec. 5). It achieves feature selection via $L_1$ regularization (known to statisticians

as the Lasso [22] and to signal processing engineers as basis pursuit [3]), so it is well-suited to learning problems with lots of data in high dimensions, such as the problem of filtering spam from your email account (Sec. 5.2). To our knowledge, no method has been proposed that reliably achieves $L_1$ regularization in large-scale problems when data is processed on-line or on-demand. Finally, it is important that we establish convergence guarantees for our method (Sec. 4). To do so, we rely on math from stochastic approximation and optimization.

## 2  Overview of algorithm

In their 1952 research paper, Robbins and Monro [15] examined the problem of tuning a control variable $x$ (e.g. amount of alkaline solution) so that the expected outcome of the experiment $F(x)$ (pH of soil) attains a desired level $\alpha$ (so your *Hydrangea* have pink blossoms). When the distribution of the experimental outcomes is unknown to the statistician or gardener, it may be still possible to take observations at $x$. In such case, Robbins and Monro showed that a particularly effective way to achieve a response level $\alpha = 0$ is to take a (hopefully unbiased) measurement $y_k \approx F(x_k)$, adjust the control variable according to

$$x_{k+1} = x_k - a_k y_k \tag{1}$$

for step size $a_k > 0$, then repeat. Provided the sequence $\{a_k\}$ behaves like the harmonic series (see Sec. 4.1), this algorithm converges to the solution $F(x^\star) = 0$.

Since the original publication, mathematicians have extended, generalized, and further weakened the convergence conditions; see [11] for some of these developments. Kiefer and Wolfowitz re-interpreted the stochastic process as one of optimizing an unconstrained objective ($F(x)$ acts as the gradient vector) and later Dvoretsky pointed out that each measurement $y$ is actually the gradient $F(x)$ plus some noise $\xi(x)$. Hence, the *stochastic gradient* algorithm. In this paper, we introduce a convergent sequence of nonlinear systems $F_\mu(x) = 0$ and interpret the Robbins-Monro process $\{x_k\}$ as solving a *constrained* optimization problem.

We focus on convex optimization problems [2] of the form

$$
\begin{array}{ll}
\text{minimize} & f(x) \\
\text{subject to} & c(x) \le 0,
\end{array}
\tag{2}
$$

where $c(x)$ is a vector of inequality constraints, $f(x)$ and $c(x)$ have continuous partial derivatives, and measurements $y_k$ of the gradient at $x_k$ are noisy. The feasible set, by contrast, should be known exactly. To simplify our exposition, we do not consider equality constraints; techniques for handling them are discussed in [13].

---

**procedure** IP–SG (Interior-point stochastic gradient)
    **for** $k = 1, 2, 3, \ldots$
        • Set max. step size $\hat{a}_k$ and centering parameter $\sigma_k$.
        • Set barrier parameter $\mu_k = \sigma_k z_k^T c(x_k)/m$.
        • Run simulation to obtain gradient observation $y_k$.
        • Compute primal-dual search direction $(\Delta x_k, \Delta z_k)$ by solving equations (6,7) with $\nabla f(x) = y_k$.
        • Run backtracking line search to find largest $a_k \le \min\{\hat{a}_k, 0.995 \min_i(-z_{k,i}/\Delta z_{k,i})\}$ such that $c(x_{k-1} + a_k \Delta x_k) < 0$, and $\min_i(\cdot)$ is over all $i$ such that $\Delta z_{k,i} < 0$.
        • Set $x_k = x_{k-1} + a_k \Delta x_k$ and $z_k = z_{k-1} + a_k \Delta z_k$.

Figure 1: Proposed stochastic gradient algorithm.

---

Convexity is a standard assumption made to simplify analysis of stochastic approximation algorithms and, besides, constrained, non-convex optimization raises unresolved complications. We assume standard constraint qualifications so we can legitimately identify optimal solutions via the Karush-Kuhn-Tucker (KKT) conditions [2, 13].

Following the standard barrier approach [7], we frame the constrained optimization problem as a sequence of unconstrained objectives. This in turn is cast as a sequence of root-finding problems $F_\mu(x) = 0$, where $\mu > 0$ controls for the accuracy of the approximate objective and should tend toward zero. As we explain, a dramatically more effective strategy is to solve for the root of the *primal-dual equations* $F_\mu(x, z)$, where $z$ represents the set of dual variables. This is the basic formula of the interior-point stochastic approximation method.

Fig. 1 outlines our main contribution. Provided $x_0$ is feasible and $z_0 > 0$, every subsequent iterate $(x_k, z_k)$ will be a feasible or "interior" point as well. Notice the absence of a sufficient decrease condition on $\|F_\mu(x, z)\|$ or suitable merit function; this is not needed in the stochastic setting. Our stochastic approximation algorithm requires a slightly non-standard treatment because the target $F_\mu(x, z)$ moves as $\mu$ changes. Fortunately, convergence under non-stationarity has been studied in the literature on tracking and adaptive filtering. The next section is devoted to deriving the primal-dual search direction $(\Delta x, \Delta z)$.

## 3 Background on interior-point methods

We motivate and derive primal-dual interior-point methods starting from the logarithmic barrier method. Barrier methods date back to the work of Fiacco and McCormick [6] in the 1960s, but they lost favour due to their unreliable nature. Ill-conditioning was long considered their undoing. However, careful analysis [7] has shown that poor conditioning is not the problem—rather, it is a deficiency in the search direction. In the next section, we exploit this very analysis to show that every iteration of our algorithm produces a stable iterate in the face of: 1) ill-conditioned linear systems, 2) noisy observations of the gradient.

The logarithmic barrier approach for the constrained optimization problem (2) amounts to solving a sequence of unconstrained subproblems of the form

$$\text{minimize} \quad f_\mu(x) \equiv f(x) - \mu \sum_{i=1}^{m} \log(-c_i(x)), \tag{3}$$

where $\mu > 0$ is the barrier parameter, and $m$ is the number of inequality constraints. As $\mu$ becomes smaller, the barrier function $f_\mu(x)$ acts more and more like the objective. The philosophy of barrier methods differs fundamentally from "exterior" penalty methods that penalize points violating the constraints [13, Chapter 17] because the logarithm in (3) prevents iterates from violating the constraints at all, hence the word "barrier".

The central thrust of the barrier method is to progressively push $\mu$ to zero at a rate which allows the iterates to converge to the constrained optimum $x^\star$. Writing out a first-order Taylor-series expansion to the optimality conditions $\nabla f_\mu(x) = 0$ about a point $x$, the Newton step $\Delta x$ is the solution to the linear equations $\nabla^2 f_\mu(x) \Delta x = -\nabla f_\mu(x)$. The barrier Hessian has long been known to be incredibly ill-conditioned—this fact becomes apparent by writing out $\nabla^2 f_\mu(x)$ in full—but an analysis by Wright [25] shows that the ill-conditioning is not harmful under the right conditions. The "right conditions" are that $x$ be within a small distance[1] from the *central path* or *barrier trajectory*, which is defined to be the sequence of isolated minimizers $x_\mu^\star$ satisfying $\nabla f_\mu(x_\mu^\star) = 0$ and $c(x_\mu^\star) < 0$. The bad news: the barrier method is ineffectual at remaining on the barrier trajectory—it pushes iterates too close to the boundary where they are no longer well-behaved [7]. Ordinarily, a convergence test is conducted for each value of $\mu$, but this is not a plausible option for the stochastic setting.

Primal-dual methods form a Newton search direction for both the primal variables and the Lagrange multipliers. Like classical barrier methods, they fail catastrophically outside the central path. But their virtue is that they happen to be extremely good at remaining on the central path (even in the stochastic setting; see Sec. 4.2). Primal-dual methods are also blessed with strong results regarding superlinear and quadratic rates of convergence [7].

The principal innovation is to introduce Lagrange multiplier-like variables $z_i \equiv -\mu/c_i(x)$. By setting $\nabla_x f_\mu(x)$ to zero, we recover the "perturbed" KKT optimality conditions:

$$F_\mu(x,z) \equiv \begin{bmatrix} \nabla_x f(x) + J^T Z \mathbf{1} \\ CZ\mathbf{1} + \mu\mathbf{1} \end{bmatrix} = 0, \tag{4}$$

where $Z$ and $C$ are matrices with $z$ and $c(x)$ along their diagonals, and $J \equiv \nabla_x c(x)$. Forming a first-order Taylor expansion about $(x, z)$, the primal-dual Newton step is the solution to

$$\begin{bmatrix} W & J^T \\ ZJ & C \end{bmatrix} \begin{bmatrix} \Delta x \\ \Delta z \end{bmatrix} = - \begin{bmatrix} \nabla_x f(x) + J^T Z \mathbf{1} \\ CZ\mathbf{1} + \mu\mathbf{1} \end{bmatrix}, \tag{5}$$

where $W = H + \sum_{i=1}^{m} z_i \nabla_x^2 c_i(x)$ is the Hessian of the Lagrangian (as written in any textbook on constrained optimization), and $H$ is the Hessian of the objective or an approximation. Through block elimination, the Newton step $\Delta x$ is the solution to the symmetric system

$$(W - J^T \Sigma J)\Delta x = -\nabla_x f_\mu(x), \tag{6}$$

where $\Sigma \equiv C^{-1} Z$. The dual search direction is then recovered according to

$$\Delta z = -(z + \mu/c(x) + \Sigma J \Delta x). \tag{7}$$

Because (2) is a convex optimization problem, we can derive a sensible update rule for the barrier parameter by guessing the distance between the primal and dual objectives [2]. This guess is typically $\mu = -\sigma z^T c(x)/m$, where $\sigma > 0$ is a centering parameter. This update is supported by the convergence theory (Sec. 4.1) so long as $\sigma_k$ is pushed to zero.

# 4 Analysis of convergence

First we establish conditions upon which the sequence of iterates generated by the algorithm converges almost surely to the solution $(x^\star, z^\star)$ as the amount of data or iteration count goes to infinity. Then we examine the behaviour of the iterates under finite-precision arithmetic.

## 4.1 Asymptotic convergence

A convergence proof from first principles is beyond the scope of this paper; we build upon the martingale convergence proof of Spall and Cristion for non-stationary systems [21].

**Assumptions:** We establish convergence under the following conditions. They may be weakened by applying results from the stochastic approximation and optimization literature.

1. **Unbiased observations:** $y_k$ is a discrete-time martingale difference with respect to the true gradient $\nabla f(x_k)$; that is, $E(y_k \mid x_k, \text{history up to time } k) = \nabla f(x_k)$.
2. **Step sizes:** The maximum step sizes $\hat{a}_k$ bounding $a_k$ (see Fig. 1) must approach zero ($\hat{a}_k \to 0$ as $k \to \infty$ and $\sum_{k=1}^{\infty} \hat{a}_k^2 < \infty$) but not too quickly ($\sum_{k=1}^{\infty} \hat{a}_k = \infty$).
3. **Bounded iterates:** $\limsup_k \|x_k\| < \infty$ almost surely.
4. **Bounded gradient estimates:** for some $\rho$ and for every $k$, $E(\|y_k\|) < \rho$.
5. **Convexity:** The objective $f(x)$ and constraints $c(x)$ are convex.
6. **Strict feasibility:** There must exist an $x$ that is strictly feasible; *i.e.* $c(x) < 0$.
7. **Regularity assumptions:** There exists a feasible minimizer $x^\star$ to the problem (2) such that first-order constraint qualification and strict complementarity hold, and $\nabla_x f(x), \nabla_x c(x)$ are Lipschitz-continuous. These conditions allow us to directly apply standard theorems on constrained optimization for convex programming [2, 6, 7, 13].

**Proposition:** Suppose Assumptions 1–7 hold. Then $\theta^\star \equiv (x^\star, z^\star)$ is an isolated (locally unique within a $\delta$-neighbourhood) solution to (2), and the iterates $\theta_k \equiv (x_k, z_k)$ of the feasible interior-point stochastic approximation method (Fig. 1) converge to $\theta^\star$ almost surely; that is, as $k$ approaches the limit, $\|\theta_k - \theta^\star\| = 0$ with probability 1.

**Proof:** See Appendix A.

## 4.2 Considerations regarding the central path

The object of this section is to establish that computing the stochastic primal-dual search direction is numerically stable. (See Part III of [23] for what we mean by "stable".) The concern is that noisy gradient measurements will lead to wildly perturbed search directions. As we mentioned in Sec. 3, interior-point methods are surprisingly stable provided the iterates remain close to the central path, but the prospect of keeping close to the path seems particularly tenuous in the stochastic setting. A key observation is that the central path is itself perturbed by the stochastic gradient estimates. Following arguments similar to those given in Sec. 5 of [7], we show that the stochastic Newton step (6,7) stays on target.

We define the *noisy central path* as $\theta(\mu, \varepsilon) = (x, z)$, where $(x, z)$ is a solution to $F_\mu(x, z) = 0$ with gradient estimate $y \equiv \nabla f(x) + \varepsilon$. Suppose we are currently at point $\theta(\mu, \varepsilon) = (x, z)$ along the path, and the goal is to move closer to $\theta(\mu^\star, \varepsilon^\star) = (x^\star, z^\star)$ by solving (5) or (6,7). One way to assess the quality of the Newton step is to compare it to the tangent line of the noisy central path at $(\mu, \varepsilon)$. Taking implicit partial derivatives at $(x, z)$, the tangent line is

$$\theta(\mu^\star, \varepsilon^\star) \approx \theta(\mu, \varepsilon) + (\mu^\star - \mu)\frac{\partial \theta(\mu, \varepsilon)}{\partial \mu} + (y^\star - y)\frac{\partial \theta(\mu, \varepsilon)}{\partial \varepsilon}, \quad \text{such that} \tag{8}$$

$$\begin{bmatrix} H & J^T \\ ZJ & C \end{bmatrix} \begin{bmatrix} (\mu^\star - \mu)\frac{\partial x}{\partial \mu} + (y^\star - y)\frac{\partial x}{\partial \varepsilon} \\ (\mu^\star - \mu)\frac{\partial z}{\partial \mu} + (y^\star - y)\frac{\partial z}{\partial \varepsilon} \end{bmatrix} = -\begin{bmatrix} y^\star - y \\ (\mu^\star - \mu)1, \end{bmatrix}. \tag{9}$$

with $y^\star \equiv \nabla f(x) + \varepsilon^\star$. Since we know that $F_\mu(x, z) = 0$, the Newton step (5) at $(x, z)$ with perturbation $\mu^\star$ and stochastic gradient estimate $y^\star$ is the solution to

$$\begin{bmatrix} H & J^T \\ ZJ & C \end{bmatrix} \begin{bmatrix} \Delta x \\ \Delta z \end{bmatrix} = -\begin{bmatrix} y^\star - y \\ (\mu^\star - \mu)1. \end{bmatrix}. \tag{10}$$

In conclusion, if the tangent line (8) is a fairly reasonable approximation to the central path, then the stochastic Newton step (10) will make good progress toward $\theta(\mu^\star, \varepsilon^\star)$.

Having established that the stochastic gradient algorithm closely follows the noisy central path, the analysis of M. H. Wright [26] directly applies, in which round-off error ($\epsilon_{\text{machine}}$) is occasionally replaced by gradient noise ($\varepsilon$). Since stability is of fundamental concern—particularly in computing the values of $W - J^T \Sigma J$, the right-hand side of (6), and the solution to $\Delta x$ and $\Delta z$—we elaborate on the significance of Wright's results in Appendix B.

## 5 On-line L1 regularization

In this section, we apply our findings to the problem of computing an $L_1$-regularized least squares estimator in an "on-line" manner; that is, by making adjustments to each new example without having to review all the previous training instances. While this problem only involves simple bound constraints, we can use it to compare our method to existing approaches such as gradient projection. We start with some background behind the $L_1$, motivate the on-line learning approach, draw some experimental comparisons with existing methods, then show that our algorithm can be used to filter spam.

Suppose we have $n$ training examples $x_i \equiv (x_{i1}, \ldots, x_{im})^T$ paired with real-valued responses $y_i$. (The notation here is separate from previous sections.) Assuming a linear model and centred coordinates, the least squares estimate $\beta$ minimizes the mean squared error (MSE). Linear regression based on the maximum likelihood estimator is one of the basic statistical tools of science and engineering and, while primitive, generalizes to many popular statistical estimators, including linear discriminant analysis [9]. Because the least squares estimator is unstable when $m$ is large, it can generalize poorly to unseen examples. The standard cure is "regularization," which introduces bias, but typically produces estimators that are better at predicting the outputs of unseen examples. For instance, the MSE with an $L_1$-penalty,

$$\text{MSE}^{(L_1)} \equiv \frac{1}{2n} \sum_{i=1}^{n} (y_i - x_i^T \beta)^2 + \frac{\lambda}{n} \|\beta\|_1, \tag{11}$$

not only prevents overfitting but tends to produce estimators that shrink many of the components $\beta_j$ to zero, resulting in sparse codes. Here, $\| \cdot \|_1$ is the $L_1$ norm and $\lambda > 0$ controls for the level of regularization. This approach has been independently studied for many problems, including statistical regression [22] and sparse signal reconstruction [3, 10], precisely because it is effective at choosing useful features for prediction.

We can treat the gradient of MSE as a sample expectation over responses of the form $-x_i(y_i - x_i^T \beta)$, so the on-line or stochastic update

$$\beta^{(\text{new})} = \beta + a x_i (y_i - x_i^T \beta), \tag{12}$$

improves the linear regression with only a single data point ($a$ is the step size).[2] This is the famed "delta rule" of Widrow and Hoff [12]. Since standard "batch" learning requires a full pass through the data for each gradient evaluation, the on-line update (12) may be the only viable option when faced with, for instance, a collection of 80 million images [16]. On-line learning for regression and classification—including $L_2$ regularization—is a well-researched topic, particularly for neural networks [17] and support vector machines (e.g. [19]). On-line learning with $L_1$ regularization, despite its ascribed benefits, has strangely avoided study. (The only known work that has approached the problem is [27] using subgradient methods.)

We derive an on-line, $L_1$-regularized learning rule of the form

$$\begin{aligned}
\beta_{\text{pos}}^{(\text{new})} &= \beta_{\text{pos}} + a\Delta\beta_{\text{pos}} \quad & z_{\text{pos}}^{(\text{new})} &= z_{\text{pos}} + a(\mu/\beta_{\text{pos}} - z_{\text{pos}} - \Delta\beta_{\text{pos}} z_{\text{pos}}/\beta_{\text{pos}}) \\
\beta_{\text{neg}}^{(\text{new})} &= \beta_{\text{neg}} + a\Delta\beta_{\text{neg}} \quad & z_{\text{neg}}^{(\text{new})} &= z_{\text{neg}} + a(\mu/\beta_{\text{neg}} - z_{\text{neg}} - \Delta\beta_{\text{neg}} z_{\text{neg}}/\beta_{\text{neg}}),
\end{aligned} \tag{13}$$

$$\begin{aligned}
\text{such that} \quad \Delta\beta_{\text{pos}} &= (x_i(y_i - x_i^T \beta) - \tfrac{\lambda}{n} + \mu/\beta_{\text{pos}})/(1 + z_{\text{pos}}/\beta_{\text{pos}}) \\
\Delta\beta_{\text{neg}} &= (-x_i(y_i + x_i^T \beta) - \tfrac{\lambda}{n} + \mu/\beta_{\text{neg}})/(1 + z_{\text{neg}}/\beta_{\text{neg}}),
\end{aligned}$$

and where $\mu > 0$ is the barrier parameter, $\beta = \beta_{\text{pos}} - \beta_{\text{neg}}$, $z_{\text{pos}}$ and $z_{\text{neg}}$ are the Lagrange multipliers associated with the lower bounds $\beta_{\text{pos}} \geq 0$ and $\beta_{\text{neg}} \geq 0$, respectively, and $a$ is a step size ensuring the variables remain in the positive quadrant. Multiplication and division in (13) are component-wise. The remainder of the algorithm (Fig. 1) consists of choosing $\mu$ and feasible step size $a$ at each iteration. Let us briefly explain how we arrived at (13).

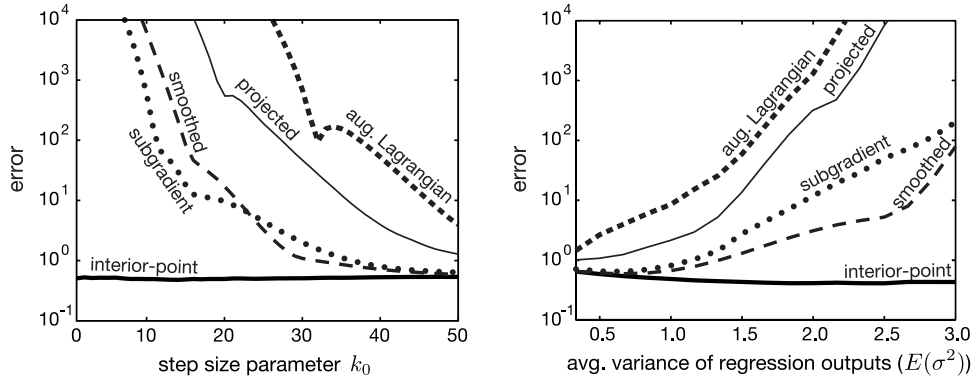

Figure 2: *(left)* Performance of constrained stochastic gradient methods for different step size sequences. *(right)* Performance of methods for increasing levels of variance in the dimensions of the training data. Note the logarithmic scale in the vertical axis.

It is difficult to find a collection of regression coefficients $\beta$ that directly minimizes $\mathrm{MSE}^{(L_1)}$ because the $L_1$ norm is not differentiable near zero. The trick is to separate the coefficients into their positive ($\beta_{\mathrm{pos}}$) and negative ($\beta_{\mathrm{neg}}$) components following [3], thereby transforming the non-smooth, unconstrained optimization problem (11) into a smooth problem with convex, quadratic objective and bound constraints $\beta_{\mathrm{pos}}, \beta_{\mathrm{neg}} \geq 0$. The regularized delta rule (13) is then obtained from direct application of the primal-dual interior-point Newton search direction (6,7) with a stochastic gradient (see Eq. 12), and identity in place of $H$.

## 5.1 Experiments

We ran four small experiments to assess the reliability and shrinkage effect of the interior-point stochastic gradient method for linear regression with $L_1$ regularization; refer to Fig. 1 and Eq. 13.[3] We also studied four alternatives to our method: 1) a subgradient method, 2) a smoothed, unconstrained approximation to (11), 3) a projected gradient method, and 4) the augmented Lagrangian approach described in [24]. See [18] for an in-depth discussion of the merits of applying the first three optimization approaches to $L_1$ regularization. All these methods have a per-iteration cost on the order of the number of features.

**Method.** For the first three experiments, we simulated 20 data sets following the procedure described in Sec. 7.5 of [22]. Each data set had $n = 100$ observations with $m = 40$ features. We defined observations by $x_{ij} = z_{ij} + z_i$, where $z_i$ was drawn from the standard normal and $z_{ij}$ was drawn i.i.d. from the normal with variance $\sigma_j^2$, which in turn was drawn from the inverse Gamma with shape 2.5 and scale $\nu = 1$. (The mean of $\sigma_j^2$ is proportional to $\nu$.) The regression coefficients were $\beta = (0, \ldots, 0, 2, \ldots, 2, 0, \ldots, 0, 2, \ldots, 2)^T$ with 10 repeats in each block [22]. Outputs were generated according to $y_i = \beta^T x_i + \epsilon$ with standard Gaussian noise $\epsilon$. Each method was executed with a single pass on the data (100 iterations) with step sizes $\hat{a}_k = 1/(k_0 + k)$, where $k_0 = 50$ by default. We chose $L_1$ penalty $\lambda/n = 1.25$, which tended to produce about 30% zero coefficients at the solution to (11). The augmented Lagrangian required a sequence of penalty terms $r_k \to 0$; after some trial and error, we chose $r_k = 50/(k_0 + k)^{0.1}$. The control variables of Experiments 1, 2 and 3 were, respectively, the step size parameter $k_0$, the inverse Gamma scale parameter $\nu$, and the $L_1$ penalty parameter $\lambda$. In Experiment 4, each example $y_i$ in the training set $x_i$ had 8 features, and we set the true coefficients were set to $\beta = (0, 0, 2, -4, 0, 0, -1, 3)^T$.

**Results.** Fig. 2 shows the results of Experiments 1 and 2, with error $\frac{1}{n}\|\beta^{\mathrm{exact}} - \beta^{\mathrm{on\text{-}line}}\|_1$ averaged over the 20 data sets, in which $\beta^{\mathrm{exact}}$ is the solution to (11), and $\beta^{\mathrm{on\text{-}line}}$ is the estimate obtained after 100 iterations of the on-line or stochastic gradient method. With a large enough step size, almost all the methods converged close to $\beta^{\mathrm{exact}}$. The stochastic interior-point method, however, always came closest to $\beta^{\mathrm{exact}}$ and, for the range of values we tried, its solution was by far the least sensitive to the step size sequence and level of variance in the observations. Experiment 3 (Fig. 3) shows that even with well-chosen step sizes for all methods,

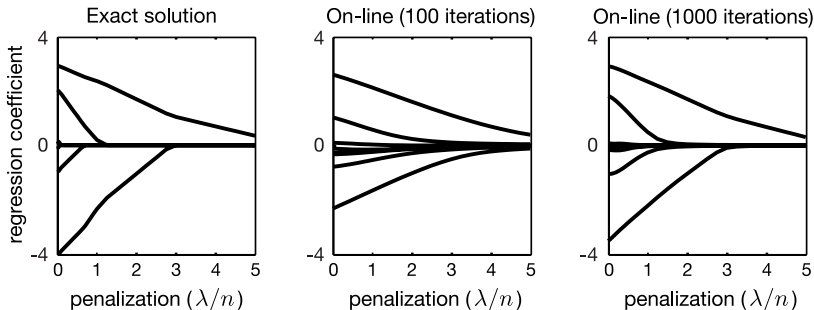

Figure 4: Shrinkage effect for different choices of the $L_1$ penalty parameter.

the stochastic interior-point method still best approximated the exact solution, and its performance did not degrade when $\lambda$ was small. (The dashed vertical line at $\lambda/n = 1.25$ in Fig. 3 corresponds to $k_0 = 50$ and $E(\sigma^2) = 2/3$ in the left and right plots of Fig. 2.) Fig. 4 shows the regularized estimates of Experiment 4. After one pass through the data *(middle)*—equivalent to a *single* iteration of an exact solver—the interior-point stochastic gradient method shrank some of the data components, but didn't quite discard irrelevant features altogether. After 10 visits to the training data *(right)*, the stochastic algorithm exhibited feature selection close to what we would normally expect from the Lasso *(left)*.

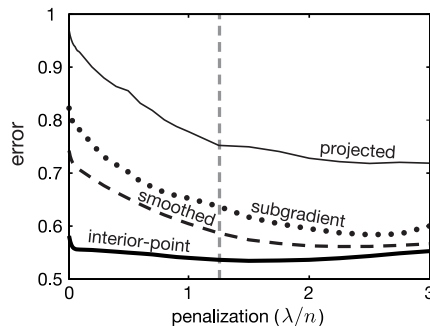

Figure 3: Performance of the methods for various choices of the $L_1$ penalty.

## 5.2 Filtering spam

Classifying email as spam or not is most faithfully modeled as an on-line learning problem in which supervision is provided *after* each email has been designated for the inbox or trash [5]. An effective filter is one that minimizes misclassification of incoming messages—throwing away a good email being considerably more deleterious than incorrectly placing a spam in the inbox. Without any prior knowledge as to what spam looks like, any filter will be error-prone at initial stages of deployment.

Spam filtering necessarily involves lots of data and an even larger number of features, so a sparse, stable model is essential. We adapted the $L_1$-regularized delta rule to the spam filtering problem by replacing the linear regression with a binary logistic regression [9]. The on-line updates are similar to (13), only $x_i^T \beta$ is replaced by $\phi(x_i^T \beta)$, with $\phi(u) \equiv 1/(1+e^{-u})$. To our knowledge, no one has investigated this approach for on-line spam filtering, though there is some work on logistic regression plus the Lasso for batch classification in text corpora [8]. Needless to say, batch learning is completely impractical in this setting.

**Method.** We simulated the on-line spam filtering task on the TREC2005 corpus [4] containing emails from the legal investigations of Enron corporation. We compared our on-line classifier ($\lambda = 10$, $\sigma = \frac{1}{2}$, $\hat{a}_i = \frac{1}{1+i}$) with two open-source software packages, SpamBayes 1.0.3 and Bogofilter 0.93.4. (These packages are publicly available at spambayes.sourceforge.net and bogofilter.sourceforge.net.) A full comparison is certainly beyond the scope of this paper; see [5] for a comprehensive evaluation. We represented each email as a vector of normalized word frequencies, and used the word tokens extracted by SpamBayes. In the end, we had an on-line learning problem involving $n = 92189$ documents and $m = 823470$ features.

| | | true | |
| --- | --- | --- | --- |
| | | not spam | spam |
| pred. | not spam | 39382 | 3291 |
| | spam | 17 | 49499 |

Results for SpamBayes

| | | true | |
| --- | --- | --- | --- |
| | | not spam | spam |
| pred. | not spam | 39393 | 5515 |
| | spam | 3 | 47275 |

Results for Bogofilter

| | | true | |
| --- | --- | --- | --- |
| | | not spam | spam |
| pred. | not spam | 39389 | 2803 |
| | spam | 10 | 49987 |

Results for Logistic + L1

Table 1: Contingency tables for on-line spam filtering task on the TREC2005 data set.

**Results.** Following [5], we use contingency tables to present results of the on-line spam filtering experiment (Table 1). The top-right/bottom-left entry of each table is the number of misclassified spam/non-spam. Everything was evaluated on-line. We tagged an email for deletion only if $p(y_i = \text{spam}) \geq 97\%$. Our spam filter dominated SpamBayes on the TREC2005 corpus, and performed comparably to Bogofilter—one of the best spam filters to date [5]. Our model's expense was slightly greater than the others. As we found in Sec. 5.1, assessing sparsity of the on-line solution is more difficult than in the exact case, but we can say that removing the 41% smallest entries of $\beta$ resulted in almost no ($< 0.001$) change.

## 6    Conclusions

Our experiments on a learning problem with noisy gradient measurements and bound constraints show that the interior-point stochastic approximation algorithm is a significant improvement over other methods. The interior-point approach also has the virtue of being much more general, and our analysis guarantees that it will be numerically stable.

**Acknowledgements.** Thanks to Ewout van den Berg, Matt Hoffman and Firas Hamze.

## Footnotes

[1]See Sec. 4.3.1 of [7] for the precise meaning of a "small distance". Since $x$ must be close to the central path but far from the boundary, the favourable neighbourhood shrinks as $\mu$ nears 0.

[2]The gradient descent direction can be a poor choice because it ignores the scaling of the problem. Much work has focused on improving the delta rule, but we shall not discuss these improvements.

[3]The MATLAB code for all our experiments is on the Web at http://www.cs.ubc.ca/~pcarbo.

## References

[1] L. Bottou and O. Bousquet, *The tradeoffs of large scale learning*, in Advances in Neural Information Processing Systems, vol. 20, 1998.

[2] S. Boyd and L. Vandenberghe, *Convex optimization*, Cambridge University Press, 2004.

[3] S. Chen, D. Donoho, and M. Saunders, *Atomic decomposition by basis pursuit*, SIAM Journal on Scientific Computing, 20 (1999), pp. 33–61.

[4] G. V. Cormack and T. R. Lynam, *Spam corpus creation for TREC*, in Proc. 2nd CEAS, 2005.

[5] ———, *Online supervised spam filter evaluation*, ACM Trans. Information Systems, 25 (2007).

[6] A. V. Fiacco and G. P. McCormick, *Nonlinear programming: sequential unconstrained minimization techniques*, John Wiley and Sons, 1968.

[7] A. Forsgren, P. E. Gill, and M. H. Wright, *Interior methods for nonlinear optimization*, SIAM Review, 44 (2002), pp. 525–597.

[8] A. Genkin, D. D. Lewis, and D. Madigan, *Large-scale Bayesian logistic regression for text categorization*, Technometrics, 49 (2007), pp. 291–304.

[9] T. Hastie, R. Tibshirani, and J. Friedman, *The elements of statistical learning*, Springer, 2001.

[10] S.-J. Kim, K. Koh, M. Lustig, S. Boyd, and D. Gorinevsky, *An interior-point method for large-scale L1-regularized least squares*, IEEE J. Selected Topics in Signal Processing, 1 (2007).

[11] H. J. Kushner and D. S. Clark, *Stochastic approximation methods for constrained and unconstrained systems*, Springer-Verlag, 1978.

[12] T. M. Mitchell, *Machine Learning*, McGraw-Hill, 1997.

[13] J. Nocedal and S. J. Wright, *Numerical Optimization*, Springer, 2nd ed., 2006.

[14] B. T. Poljak, *Nonlinear programming methods in the presence of noise*, Mathematical Programming, 14 (1978), pp. 87–97.

[15] H. Robbins and S. Monro, *A stochastic approximation method*, Annals Math. Stats., 22 (1951).

[16] B. C. Russell, A. Torralba, K. P. Murphy, and W. T. Freeman, *LabelMe: a database and web-based tool for image annotation*, Intl. Journal of Computer Vision, 77 (2008), pp. 157–173.

[17] D. Saad, ed., *On-line learning in neural networks*, Cambridge University Press, 1998.

[18] M. Schmidt, G. Fung, and R. Rosales, *Fast optimization methods for L1 regularization*, in Proceedings of the 18th European Conference on Machine Learning, 2007, pp. 286–297.

[19] S. Shalev-Shwartz, Y. Singer, and N. Srebro, *Pegasos: primal estimated sub-gradient solver for SVM*, in Proceedings of the 24th Intl. Conference on Machine learning, 2007, pp. 807–814.

[20] J. C. Spall, *Adaptive stochastic approximation by the simultaneous perturbation method*, IEEE Transactions on Automatic Control, 45 (2000), pp. 1839–1853.

[21] J. C. Spall and J. A. Cristion, *Model-free control of nonlinear stochastic systems with discrete-time measurements*, IEEE Transactions on Automatic Control, 43 (1998), pp. 1148–1210.

[22] R. Tibshirani, *Regression shrinkage and selection via the Lasso*, Journal of the Royal Statistical Society, 58 (1996), pp. 267–288.

[23] L. N. Trefethen and D. Bau, *Numerical linear algebra*, SIAM, 1997.

[24] I. Wang and J. C. Spall, *Stochastic optimization with inequality constraints using simultaneous perturbations and penalty functions*, in Proc. 42nd IEEE Conf. Decision and Control, 2003.

[25] M. H. Wright, *Some properties of the Hessian of the logarithmic barrier function*, Mathematical Programming, 67 (1994), pp. 265–295.

[26] ———, *Ill-conditioning and computational error in interior methods for nonlinear programming*, SIAM Journal on Optimization, 9 (1998), pp. 84–111.

[27] A. Zheng, *Statistical software debugging*, PhD thesis, University of California, Berkeley, 2005.
